# Bayesian Pedigree Analysis using Measure Factorization

**Alexandre Bouchard-Côté**
Statistics Department
University of British Columbia
bouchard@stat.ubc.ca

**Bonnie Kirkpatrick**
Computer Science Department
University of British Columbia
bbkirk@cs.ubc.ca

## Abstract

Pedigrees, or family trees, are directed graphs used to identify sites of the genome that are correlated with the presence or absence of a disease. With the advent of genotyping and sequencing technologies, there has been an explosion in the amount of data available, both in the number of individuals and in the number of sites. Some pedigrees number in the thousands of individuals. Meanwhile, analysis methods have remained limited to pedigrees of $< 100$ individuals which limits analyses to many small independent pedigrees.

Disease models, such those used for the linkage analysis log-odds (LOD) estimator, have similarly been limited. This is because linkage analysis was originally designed with a different task in mind, that of ordering the sites in the genome, before there were technologies that could reveal the order. LODs are difficult to interpret and nontrivial to extend to consider interactions among sites. These developments and difficulties call for the creation of modern methods of pedigree analysis.

Drawing from recent advances in graphical model inference and transducer theory, we introduce a simple yet powerful formalism for expressing genetic disease models. We show that these disease models can be turned into accurate and computationally efficient estimators. The technique we use for constructing the variational approximation has potential applications to inference in other large-scale graphical models. This method allows inference on larger pedigrees than previously analyzed in the literature, which improves disease site prediction.

## 1 Introduction

Finding genetic correlates of disease is a long-standing important problem with potential contributions to diagnostics and treatment of disease. The pedigree model for inheritance is one of the best defined models in biology, and it has been an area of active statistical and biological research for over a hundred years.

The most commonly used method to analyze genetic correlates of disease is quite old. After Mendel introduced, in 1866, the basic model for the inheritance of genomic sites [1] Sturtevant was the first, in 1913, to provide a method for ordering the sites of the genome [2]. The method of Sturtevant became the foundation for linkage analysis with pedigrees [3, 4, 5, 6]. The problem can be thought of in Sturtevant's framework as that of finding the position of a disease site relative to an map of existing sites. This is the log-odds (LOD) estimator for linkage analysis which is a likelihood ratio test, described in more detail below.

The genomic data available now is quite different than the type of data available when LOD was initially developed. Genomic sites are becoming considerably denser in the genome and technologies allow us to interrogate the genome for the position of sites [7]. Additionally, most current pedigree

analysis methods are exponential either in the number of sites or in the number of individuals. This produces a limit on the size of the pedigrees under consideration to around $< 100$ individuals. This is in contrast to the size of pedigrees being collected: for example the work of [8] includes a connected human pedigree containing 13 generations and 1623 individuals, and the work of [9] includes a connected non-human data set containing thousands of breeding dogs. Apart from the issues of pedigree size, the LOD value is difficult to interpret, since there are few models for the distribution of the statistic. These developments and difficulties call for the creation of modern methods of pedigree analysis.

In this work, we propose a new framework for expressing genetic disease models. The key component of our models, the Haplotype-Phenotype Transducer (HPT), draws from recent advances in graphical model inference and transducer theory [10], and provides a simple and flexible formalism for building genetic disease models. The output of inference over HPT models is a posterior distribution over disease sites, which is easier to interpret than LOD scores.

The cost of this modeling flexibility is that the graphical model corresponding to the inference problem is larger and has more loops that traditional pedigree graphical models. Our solution to this challenge is based on the observation that the difficult graphical model can be covered by a collection of tractable forest graphical models. We use a method based on measure factorization [11] to efficiently combine these approximations. Our approach is applicable to other dense graphical models, and we show that empirically it gives accurate approximations in dense graphical models containing millions of nodes as well as short and long cycles. Our approximation can be refined by adding more trees in the forest, with a cost linear in the number of forests used in the cover. We show that considerable gains in accuracy can be obtained this way. In contrast, methods such as [12] can suffer from an exponential increase in running time when larger clusters are considered.

Our framework can be specialized to create analogues of classical penetrance disease models [13]. We focus on these special cases here to compare our method with classical ones. Our experiments show that even for these simpler cases, our approach can achieve significant gains in disease site identification accuracy compared to the most commonly used method, Merlin's implementation of LOD scores [3, 5]. Moreover, our inference method allows us to perform experiments on unprecedented pedigree sizes, well beyond the capacity of Merlin and other pedigree analysis tools typically used in practice.

While graphical models have played an important role in the development of pedigree analysis methods [14, 15], only recently were variational methods applied to the problem [6]. However this previous work is based on the same graphical model as classical LOD methods, while ours significantly differs.

Most current work on more advanced disease models have focused on a very different type of data, population data, for genome wide association studies (GWAS) [16]. Similarly, state of the art work on the related task of imputation generally makes similar population assumptions [17].

## 2  Background

Every individual has two copies of each chromosome, one copy is a collage of the mother's two chromosomes while the other is a collage of the father's two chromosomes. The point at which the copying of the chromosomes switches from one of the grand-maternal (grand-paternal) chromosomes to the other, is called a *recombination breakpoint*. A *site* is a particular position in the genome at which we can obtain measurable values. For the purposes of this paper, an *allele* is the nucleotide at a particular site on a particular chromosome. A *haplotype* is the sequence of alleles that appear together on the same chromosome.

If we had complete data, we would know the positions of all of the haplotypes, all of the recombination breakpoints as well as which allele came from which parent. This information is not obtainable from any known experiment. Instead, we have *genotype* data which is the *set* of nucleotides that appear in an individual's genome at a particular site. Given that the genotype is a set, it is unordered, and we do not know which allele came from which parent. All of this and the recombination breakpoints must be inferred. An example is given in the Supplement.

A *pedigree* is a directed acyclic graph with individuals as nodes, where boxes are males and circles are females, and edges directed downward from parent to child. Every individual must have either no parents or one parent of each gender. The individuals without parents in the graph are called *founders*, and the individuals with parents are *non-founders*. The pedigree encodes a set of relationships that constrain the allowed inheritance options. These inheritance options define a probability distribution which is investigated during pedigree analysis.

Assume a single-site disease model, where a diploid genotype, $G_D$, determines the affection status (phenotype), $P \in \{$'h','d'$\}$, according to the penetrance probabilities: $f_2 = \mathbb{P}(P = $'d'$|G_D = 11)$, $f_1 = \mathbb{P}(P = $'d'$|G_D = 10)$, $f_0 = \mathbb{P}(P = $'d'$|G_D = 00)$. Here the disease site usually has a disease allele, 1, that confers greater risk of having the disease. For convenience, we denote the penetrance vector as $f = (f_2, f_1, f_0)$.

Let the pedigree model for $n$ individuals be specified by a pedigree graph, a disease model $f$, and the minor allele frequency, $\mu$, for a single site of interest, $k$. Let $P = (P_1, P_2, ..., P_n)$ be a vector containing the affection status of each individual. Let $G = (G_1, G_2, ..., G_n)$ be the genotype data for each individual. Between the disease site and site $k$, we model the per chromosome, per generation recombination fraction, $\rho$, which is the frequency with which recombinations occur between those two sites. Other sites linked to $k$ can contribute to our estimate via their arrangement in single first-order Markov chain with some sites falling to the left of the disease site and others to the right of the site of interest. Previous work has shown that given a pedigree model, affection data, and genotype data, we can estimate $\rho$.

We define the likelihood as $L(\rho) = \mathbb{P}(P = p, G = g|\rho, f, \mu)$ where $\rho$ is the recombination probability between the disease site and the first site, $p$ are the founder allele frequencies, and $f$ are the penetrance probabilities. To test for linkage between the disease site and the other sites, we maximize the likelihood to obtain the optimal recombination fraction $\rho^* = \text{argmax}_\rho L(\rho)/L(1/2)$. The test we use is the likelihood ratio test where the null hypothesis is that of no linkage ($\rho = 1/2$). Generally referred to as the log-odd score (or LOD score), the log of this likelihood ratio is $\log L(\rho^*) - \log L(1/2)$.

# 3 Methods

In this section, we describe our model for inferring relationships between phenotypes and genotyped pedigree datasets. We start by giving a high-level description of the generative process.

The first step in this generative process consists in sampling a collection of disease model (DM) variables, which encode putative relationships between the genetic sites and the observed phenotypes. There is one disease model variable for each site, $s$, and to a first approximation, $D_s$ can be thought as taking values zero or one, depending on whether site $s$ is the closest to the primary genetic factor involved in a disease (a more elaborate example is presented in the Supplement). We use $\mathcal{C}$ to denote the values $D_s$ can take.

The second generative step consists in sampling the chromosomes or haplotypes of a collection of related individuals. We denote these variables by $H_{i,s,x}$, where, from now on, $i$ is used to index individuals, $s$, to index sites, and $x \in \{$ 'father', 'mother' $\}$, to index chromosome parental origin. For SNP data, the set of values $\mathcal{H}$ that $H_{i,s,x}$ can take generally contains two elements (alleles). A related variable, the inheritance variables $R_{i,s,x}$, will be sampled jointly with the $H_{i,s,x}$'s to keep track of the grand-parental origin of each chromosome segment. See Figure 1(a) for a factor graph representation of the random variables.

Finally, the phenotype $P_i$, which we assume is taken from a finite set $\mathcal{P}$, can be sampled for each individual $i$ in the pedigree. We will define the distribution of $P_i$ conditionally on the haplotype of the individual in question, $H_i$, and on the global disease model $D$. Note that variables with missing indices are used to denote random vectors or matrices, for example $D = (D_1, \ldots, D_S)$, where $S$ denotes the number of sites.

To summarize this high-level view of the process, and to introduce notations for the distributions involved:

$$D \sim \text{DM}(\cdot)$$
$$R_i \sim \text{Recomb}(\cdot) \quad \text{for all } i$$

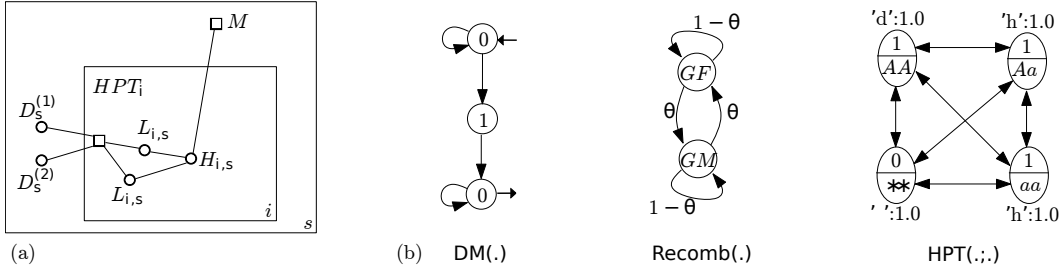

Figure 1: (a) The pedigree graphical model for independent sites. There are two plates, one for each individual and one for each site. The nodes are labeled as follows: $M$ for the marriage node which enforces the Mendelian inheritance constraints, $H$ for haplotype, $L$ and $L'$ for the two alleles, $D^{(1)}$ for the disease site indicator, and $D^{(2)}$ for the disease allele value. (b) The transducer for $\text{DM}(\cdot)$ has three nodes with the start node indicated by an in-arrow and the end node indicated by an out-arrow. The transducer for $\text{Recomb}(\cdot)$ has recombination parameter $\theta$. This assumes a constant recombination rate across sites, but non-constant rates can be obtained with a bigger automaton. This transducer for $\text{HPT}(\cdot)$ models a recessive disease where the input at each state is the disease (top) and haplotype alleles (bottom). For these last two transducers any node can be the start or end node.

The remaining variables (the non-founder individuals' haplotype variables) are obtained deterministically from the values of the founders and the inheritance: $H_{i,s,x} = H_{x(i),s,R_{i,s,x}}$, where $x(i)$ denotes the index of the father (mother) of $i$ if $x = $ 'father' ('mother'). The distribution on the founder haplotypes is a product of independent Bernoulli distributions, one for each site (the parameters of these Bernoulli distributions is not restricted to be identically distributed and can be estimated [3]). Each genotype variable $G_s$ is obtained via a deterministic function of $H$. Having generated all the haplotypes and disease variables, we denote the conditional distribution of the phenotypes as follows:

$$P_i|(D, H_i) \sim \text{HPT}(\,\cdot\,; D, H_i),$$

where HPT stands for a *Haplotype-Phenotype Transducer*.

We now turn to the description of these distributions, starting with the most important one, $\text{HPT}(\,\cdot\,; D, H_i)$. Formally, this distribution on phenotypes is derived from a weighted automaton, where we view the vectors $D$ and $H_i$ as an input string of length $S$, the $s$-th character of which is the triplet $(D_s, H_{i,s,\text{'father'}}, H_{i,s,\text{'mother'}})$. We view each of the sampled phenotypes as a length-one output from a weighted transducer given the input $D, H_i$. Longer outputs could potentially be used for more complex phenotypes or diseases.

To illustrate this construction, we show that classical, Mendelian models such as recessive phenotypes are a special case of this formalism. We also make two simplifications to facilitate exposition: first, that the disease site is one of the observed sites, and second, that the disease allele is the less frequent (minor) allele (we show in the Supplement a slightly more complicated transducer that does not make these assumptions).

Under the two above assumptions, we claim that the state diagrams in Figure 1(b) specify an HPT transducer for a recessive disease model. Each oval corresponds to a hidden transducer state, and the annotation inside the oval encodes the tuple of input symbols that the corresponding state consumes. The emission is depicted on top of the states, with for example 'd': 1.0 denotes that a disease indicator is emitted with weight one. We use 'h' for the non-disease (healthy) indicator, and $\epsilon$ for the null emission.

The probability mass function of the HPT is defined as:

$$\text{HPT}(p; c, h) = \frac{\sum_{z \in \mathcal{Z}_{\text{HPT}}(h, c \to p)} w_{\text{HPT}}(z)}{\sum_{z' \in \mathcal{Z}_{\text{HPT}}(h, c \to \star)} w_{\text{HPT}}(z')},$$

where $h \in \mathcal{H}^S, c \in \mathcal{C}^S, p \in \mathcal{P}$, and $\mathcal{Z}_{\text{HPT}}(h, c \to p)$ denotes the set of *valid paths* in the space $\mathcal{Z}$ of hidden states. The valid paths are sequences of hidden states (depicted by black circles in Figure 1(b)) starting at the source and ending at the sink, consuming $c, h$ and emitting $p$ along the way. The star in the denominator of the above equation is used to denote unconstrained emissions.

In other words, the denominator is the normalization of the weighted transducer [10]. The set of valid paths is implicitly encoded in the transition diagram of the transducer, and the weight function $w_{\mathrm{HPT}} : \mathcal{Z}^* \to [0, \infty)$ can similarly be compactly represented by only storing weights for individual transitions and multiplying them to get a path weight.

The set of valid paths along with their weights can be thought of as encoding a parametric disease model. For example, with a recessive disease, shown in Figure 1(b), we can see that if the transducer is at the site of the disease (encoded as the current symbol in $c$ being equal to 1) then only an input homozygous haplotype 'AA' will lead to an output disease phenotype 'd.' This formalism gives a considerable amount of flexibility to the modeler, who can go beyond simple Mendelian disease models by constructing different transducers.

The DM distribution is defined using the same machinery as for the HPT distribution. We show in Figure 1(b) a weighted automaton that encodes the prior that exactly one site is involved in the disease, with an unknown, uniformly distributed location in the genome. The probability mass function of the distribution is given by:

$$\mathrm{DM}(c) = \frac{\sum_{z \in \mathcal{Z}_{\mathrm{DM}}(\to c)} w_{\mathrm{DM}}(z)}{\sum_{z' \in \mathcal{Z}_{\mathrm{DM}}(\to \star)} w_{\mathrm{DM}}(z')},$$

where $\mathcal{Z}_{\mathrm{DM}}(\to c)$ and $\mathcal{Z}_{\mathrm{DM}}(\to \star)$ are direct analogues to the HPT case, with the difference being that no input is read in the DM case.

The last distribution in our model, Recomb, is standard, but we present it in the new light of the transducer formalism. Refer to Figure 1(b) for an example based on the standard recombination model derived from the marginals of a Poisson process. We use the analogous notation:

$$\mathrm{Recomb}(r) = \frac{\sum_{z \in \mathcal{Z}_{\mathrm{Recomb}}(\to r)} w_{\mathrm{Recomb}}(z)}{\sum_{z' \in \mathcal{Z}_{\mathrm{Recomb}}(\to \star)} w_{\mathrm{Recomb}}(z')}.$$

## 4   Computational Aspects

Probabilistic inference in our model is computationally challenging: the variables $L, H$ alone induce a loopy graph [18], and the addition of the variables $D, P$ introduces more loops as well as deterministic constraints, which further complicates the situation. After explaining in more detail the graphical model of interest, we discuss in this section the approximation algorithm that we have used to infer haplotypes, disease loci, and other disease statistics.

We show in Figure 1(a) the factor graph obtained after turning the observed variables (genotypes and phenotypes) into potentials (we show a more detailed version in the Supplement). We have also taken the pointwise product of potentials whenever possible (in the case of the transducer potentials, how this pointwise product is implemented is discussed in [10]). Note that our graphical model has more cycles than standard pedigree graphical models [19]; even if we assumed the sites to be independent and the pedigree to be acyclic, our graphical model would still be cyclic.

Our inference method is based on the following observation: if we kept only one subtype of factors in the Supplement, say only those connected to the recombination variables $R$, then inference could be done easily. More precisely, inference would reduce to a collection of small, standard HMMs inference problems, which can be done using existing software.

Similarly, by covering the pedigree graph with a collection of subtrees, and removed the factors for disease and recombination, we can get a collection of acyclic pedigrees, one for each site, and hence a tractable problem (the sum-product algorithm in this case is called the Elston-Stewart algorithm [14] in the pedigree literature).

We are therefore in a situation where we have several restricted views on our graphical model yielding efficiently solved subproblems. How to combine the solutions of these tractable subproblems is the question we address in the remainder of this section.

The most common way this is approached, in pedigrees [20] and elsewhere [21], is via block Gibbs sampling. However, block Gibbs sampling does not apply readily to our model. The main difficulty arises when attempting to resample $D$: because of the deterministic constraints that arise even in

the simplest disease model, it is necessary to sample $D$ in a block also containing a large subset of $R$ and $H$. However this cannot be done efficiently since $D$ is connected to all individuals in the pedigree. More formally, the difficulty is that some of the components we wish to resample are $b$-acyclic (*barely acyclic*) [22]. Another method, closer to ours, is the EP algorithm of [23], which however considers a single tree approximant, while we can accommodate several at once. As we show in the empirical section, it is advantageous to do so in pedigrees.

An important feature that we will exploit in the development of method is the *forest cover property* of the tractable subproblems: we view each tractable subproblem as a subgraph of the initial factor graph, and ask that the union of these subgraph coincides with the original factor graph.

Previous variational approaches have been proposed to exploit such forest covers. The most well-known example, the structured mean field approximation, is unfortunately non-trivial to optimize in the $b$-acyclic case [22]. Tree reweighted belief propagation [24] has an objective function derived from a forest distribution, however the corresponding algorithms are based on local message passing rather than large subproblems.

We propose an alternative based on the measure factorization framework [11]. As we will see, this yields an easy to implement variation approximation that can efficiently exploit arbitrary forest cover approximations. Since the measure factorization interpretation of our approach is not specific to pedigrees, we present it in the context of a generic factor graph over a discrete space, viewed as an exponential family with sufficient statistics $\phi$, log normalization $A$, and parameters $\theta$:

$$\mathbb{P}(X = x) = \exp\left\{ \langle \phi(x), \theta \rangle - A(\theta) \right\}. \tag{1}$$

To index the factors, we use $\varphi \in \mathcal{F} = \{1, ..., F\}$, and $v$ to index the $V$ variables in the factor graph.

We start by reparameterizing the exponential family in terms of a larger vector $y$ of variables. Let us also denote the number of nodes connected to factor $\varphi$ by $n_\varphi$. This vector $y$ has $N = \sum_\varphi n_\varphi$ components, each corresponding to a pair containing a factor and a node index attached to it, and denoted by $y_{\varphi,v}$. The reparameterization is given by:

$$\mathbb{P}(Y = y) = \exp\left\{ \langle \phi(y), \theta \rangle - A'(\theta) \right\} \prod_{\varphi,\varphi' \in \mathcal{F}} \prod_v \mathbf{1}[y_{\varphi,v} = y_{\varphi',v}]. \tag{2}$$

Because of the indicator variables in the right hand side of Equation 2, the set of $y$'s with $\mathbb{P}(Y = y) > 0$ is in bijection with the set of $x$'s with $\mathbb{P}(X = x) > 0$. It is therefore well-defined to overload the variable $\phi$ in the same equation. Similarly, we have that $A' = A$. This reparameterization is inspired by the auxiliary variables used to construct the sampler of Swendsen-Wang [25].

Next, suppose that the sets $\mathcal{F}_1, \ldots, \mathcal{F}_K$ form a forest cover of the factor graph, $\mathcal{F}_k \subset \mathcal{F}$. Then, for $k \in \{1, \ldots, K\}$, we build as follows the super-partitions required for the measure factorization to apply (as defined in [11]):

$$A_k(\theta) = \sum_y \exp\left\{ \langle \phi(y), \theta \rangle \right\} \prod_{\varphi,\varphi' \in \mathcal{F}_k} \prod_v \mathbf{1}[y_{\varphi,v} = y_{\varphi',v}]. \tag{3}$$

Note that computing each $A_k$ is tractable: it corresponds to computing the normalization of one of the forest covering the graphical model. Similarly, gradients of $A_k$ can be computed as the moments of a tree shaped graphical model. Also, the product over $k$ of the base measures in Equation 3 is equal to the base measure of Equation 2. We have therefore constructed a valid measure factorization. With this construction in hand, it is then easy to apply the measure factorization framework to get a principled way for the different subproblem views to exchange messages [11].

## 5 Experiments

We did two sets of experiments. Haplotype reconstructions were used to assess the quality of the variational approximation. Disease predictions were used to validate the HPT disease model.

**Simulations.** Pedigree graphs were simulated using a Wright-Fisher model [26]. In this model there is a fixed number of male individuals, $n$, and female individuals, $n$, per generation, making the population size $2n$. The pedigree is built starting from the oldest generation. Each successively more recent generation is built by having each individual in that generation choose uniformly at random one female parent and one male parent. Notice that this process allows inbreeding.

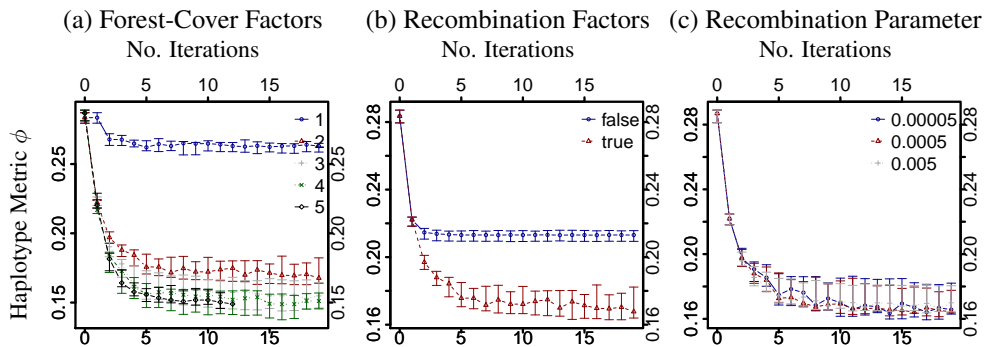

Figure 2: The pedigree was generated with the following parameters, number of generations 20 and $n = 15$ which resulted in a pedigree with 424 individuals, 197 marriage nodes, 47 founders. We simulated 1000 markers. The metric used for all panels is the haplotype reconstruction metric. Panel (a) shows the effect of removing factors from the forest cover of the pedigree where the lines are labeled with the number of factors that each experiment contains. Panel (b) shows the effect of removing the recombination factor (false) or using it (true). Together, panels (a-b) show that having more factors helps inference. Panel (c) shows the effect of an incorrect recombination parameter on inference. The correct parameter, with which the data was generated, is line 0.0005. Two incorrect parameters are shown 0.00005 and 0.005. This panel shows that the recombination parameter can be off by an order of magnitude and the haplotype reconstruction is robust.

Genotype data were simulated in the simulated pedigree graph. The founder haplotypes were drawn from an empirical distribution (see Supplement for details). The recombination parameters used for inheritance are given in the Supplement. We then simulated the inheritance and recombination process to obtain the haplotypes of the descendants using the external program [27]. We used two distributions for the founder haplotypes, corresponding to two data sets.

Individuals with missing data were sampled, where each individual either has all their genetic data missing or not. A random 50% of the non-founder individuals have missing data. An independent 50% of individuals have missing phenotypes for the disease prediction comparison.

**Haplotype Reconstruction.** For the haplotype reconstruction, the inference being scored is, for each individual, the maximum *a posteriori* haplotype predicted by the marginal haplotype distribution. These haplotypes are not necessarily Mendelian consistent, meaning that it is possible for a child to have an allele on the maternal haplotype that could not possibly be inherited from the mother according to the mother's marginal distribution. However, transforming the posterior distribution over haplotypes into a set of globally consistent haplotypes is somewhat orthogonal to the methods in this paper, and there exist methods for this task [28]. The goal of this comparison is threefold: 1) to see if adding more factors improves inference, 2) to see if more iterations of the measure factorization algorithm help, and 3) to see if there is robustness of the results to the recombination parameters.

Synthetic founder haplotypes were simulated, see Supplement for details. Each experiment was replicated 10 times where for each replicate the founder haplotypes were sampled with a different random seed. We computed a metric $\phi$ which is a normalized count of the number of sites that differ between the held-out haplotype and the predicted haplotype. See the Supplement for details.

Figure 2 shows the results for the haplotype reconstruction. Panels (a) and (b) show that adding more factors helps inference accuracy. Panel (c) shows that inference accuracy is robust to an incorrect recombination parameter.

**Disease Prediction.** For disease prediction, the inference being scored is the ranking of the sites given by our Bayesian method as compared with LOD estimates computed by Merlin [3]. The disease models we consider are recessive $f = (0.95, 0.05, 0.05)$ and dominant $f = (0.95, 0.95, 0.05)$. The disease site is one of the sites chosen uniformly at random. The goal of this comparison is to see whether our disease model performs at least as well as the LOD estimator used by Merlin.

| Pedigree | | | Disease model | | | HPT | | LOD [3] | |
|---|---|---|---|---|---|---|---|---|---|
| Generations | Leaves | Individuals | $f_2$ | $f_1$ | $f_0$ | Mean $\psi$ | SD $\psi$ | Mean $\psi$ | SD $\psi$ |
| 3 | 8 | 22 | 0.95 | 0.05 | 0.05 | 0.08 | (0.09) | 0.25 | (0.20) |
|  | 10 | 25 |  |  |  | 0.07 | (0.09) | 0.52 | (0.44) |
|  | 12 | 34 |  |  |  | 0.04 | (0.04) | 0.45 | (0.23) |
| 3 | 6 | 16 |  |  |  | 0.04 | (0.05) | 0.27 | (0.31) |
| 4 |  | 20 |  |  |  | 0.08 | (0.09) | 0.35 | (0.31) |
| 5 |  | 24 |  |  |  | 0.14 | (0.16) | 0.20 | (0.22) |
| 5 | 100 | 418 |  |  |  | 1e-3 | (2e-3) | Out of memory | |
|  | 200 | 882 |  |  |  | 4e-4 | (1e-3) | Out of memory | |
|  | 300 | 1276 |  |  |  | 6e-4 | (1e-3) | Out of memory | |
| 3 | 8 | 22 | 0.95 | 0.95 | 0.05 | 0.14 | (0.15) | 0.22 | (0.23) |
|  | 10 | 25 |  |  |  | 0.11 | (0.14) | 0.33 | (0.40) |
|  | 12 | 34 |  |  |  | 0.12 | (0.22) | 0.22 | (0.16) |

Table 1: This table gives the performance of our method and Merlin for recessive and dominant diseases as measured by the disease prediction metric. The sizes of the simulated pedigrees are given in the first three columns, the disease model in the next three columns, and the performance of our method and that of Merlin in the final four columns. In all instances, our method outperforms Merlin sometimes by an order of magnitude. Results suggest that the standard deviation of our method is smaller than that of Merlin. Notably, Merlin cannot even analyze the largest pedigrees, because Merlin does exact inference.

The founder haplotypes were taken from the phased haplotypes of the JPT+CHB HapMap [29] populations, see Supplement for details. Each experiment was replicated 10 times where for each replicate the founder haplotypes were sampled with a different random seed. We computed a metric $\psi$ which is roughly the rank of the disease site in the sorted list of predictions given by each method.

Table 1 compares the performance of our method against that of Merlin. In every case our method has better accuracy. The results suggest that our method has a lower standard deviation. Within each delineated row of the table, the mean $\psi$ are not comparable because the pedigrees might be of different complexities. Between delineated rows of the table, we can compare the effect of pedigree size, and we observe that larger pedigrees aid in disease site prediction. Indeed, the largest pedigree of 1276 individuals reaches an accuracy of $6e^{-4}$. This pedigree is the largest pedigree that we know of being analyzed in the literature.

# 6 Discussion

This paper introduces a new disease model and a new variational inference method which are applied to find a Bayesian solution to the disease-site correlation problem. This is in contrast to traditional linkage analysis where a likelihood ratio statistic is computed to find the position of the disease site relative to a map of existing sites. Instead, our approach is to use a Haplotype-Phenotype Transducer to obtain a posterior for the probability of each site to be the disease site. This approach is well-suited to modern data which is very dense in the genome. Particularly with sequencing data, it is likely that either the disease site or a nearby site will be observed.

Our method performs well in practice both for genotype prediction and for disease site prediction. In the presence of missing data, where for some individuals the whole genome is missing, our method is able to infer the missing genotypes with high accuracy. As compared with LOD linkage analysis method, our method was better able to predict the disease site when one observed site was responsible for the disease.

# References

[1] G. Mendel. *Experiments in plant-hybridisation*. In English Translation and Commentary by R. A. Fisher, J.H. Bennett, ed. Oliver and Boyd, Edinburgh 1965, 1866.

[2] A. H. Sturtevant. The linear arrangement of six sex-linked factors in drosophila, as shown by their mode of association. *Journal of Experimental Zoology*, 14:43–59, 1913.

[3] GR Abecasis, SS Cherny, WO Cookson, et al. Merlin-rapid analysis of dense genetic maps using sparse gene flow trees. *Nature Genetics*, 30:97–101, 2002.

[4] M Silberstein, A. Tzemach, N. Dovgolevsky, M. Fishelson, A. Schuster, and D. Geiger. On-line system for faster linkage analysis via parallel execution on thousands of personal computers. *Americal Journal of Human Genetics*, 78(6):922–935, 2006.

[5] D. Geiger, C. Meek, and Y. Wexler. Speeding up HMM algorithms for genetic linkage analysis via chain reductions of the state space. *Bioinformatics*, 25(12):i196, 2009.

[6] C. A. Albers, M. A. R. Leisink, and H. J. Kappen. The cluster variation method for efficient linkage analysis on extended pedigrees. *BMC Bioinformatics*, 7(S-1), 2006.

[7] M. L. Metzker. Sequencing technologies–the next generation. *Nat Rev Genet*, 11(1):31–46, January 2010.

[8] M. Abney, C. Ober, and M. S. McPeek. Quantitative-trait homozygosity and association mapping and empirical genome wide significance in large, complex pedigrees: Fasting serum-insulin level in the hutterites. *American Journal of Human Genetics*, 70(4):920 – 934, 2002.

[9] N.B. Sutter and et al. A Single IGF1 Allele Is a Major Determinant of Small Size in Dogs. *Science*, 316(5821):112–115, 2007.

[10] M. Mohri. *Handbook of Weighted Automata*, chapter 6. Monographs in Theoretical Computer Science. Springer, 2009.

[11] A. Bouchard-Côté and M. I. Jordan. Variational Inference over Combinatorial Spaces. In *Advances in Neural Information Processing Systems 23 (NIPS)*, 2010.

[12] J. S. Yedidia, W. T. Freeman, and Y. Weiss. Bethe free energy, Kikuchi approximations and belief propagation algorithms. In *Advances in Neural Information Processing Systems (NIPS)*, 2001.

[13] E. M. Wijsman. *Penetrance*. John Wiley & Sons, Ltd, 2005.

[14] R.C. Elston and J. Stewart. A general model for the analysis of pedigree data. *Human Heredity*, 21:523–542, 1971.

[15] E.S. Lander and P. Green. Construction of multilocus genetic linkage maps in humans. *Proceedings of the National Academy of Science*, 84(5):2363–2367, 1987.

[16] J. Marchini, P. Donnelly, and L. R. Cardon. Genome-wide strategies for detecting multiple loci that influence complex diseases. *Nat. Genet.*, 37(4):413–417, 2005.

[17] Y. W. Teh, C. Blundell, and L. T. Elliott. Modelling genetic variations with fragmentation-coagulation processes. In *Advances In Neural Information Processing Systems*, 2011.

[18] A. Piccolboni and D. Gusfield. On the complexity of fundamental computational problems in pedigree analysis. *Journal of Computational Biology*, 10(5):763–773, 2003.

[19] S. L. Lauritzen and N. A. Sheehan. Graphical models for genetic analysis. *Statistical Science*, 18(4):489–514, 2003.

[20] A. Thomas, A. Gutin, V. Abkevich, and A. Bansal. Multilocus linkage analysis by blocked Gibbs sampling. *Statistics and Computing*, 10(3):259–269, July 2000.

[21] G. O. Roberts and S. K. Sahu. Updating schemes, correlation structure, blocking and parameterization for the Gibbs sampler. *Journal of the Royal Statistical Society: Series B (Statistical Methodology)*, 59(2):291–317, 1997.

[22] A. Bouchard-Côté and M.I. Jordan. Optimization of structured mean field objectives. In *Proceedings of the Twenty-Fifth Conference Annual Conference on Uncertainty in Artificial Intelligence (UAI-09)*, pages 67–74, Corvallis, Oregon, 2009. AUAI Press.

[23] T. Minka and Y. Qi. Tree-structured approximations by expectation. In *Advances in Neural Information Processing Systems (NIPS)*, 2003.

[24] M. J. Wainwright, T. S. Jaakkola, and A. S. Willsky. Tree-reweighted belief propagation algorithms and approximate ML estimation by pseudo-moment matching. In *AISTATS*, 2003.

[25] R. H. Swendsen and J.-S. Wang. Nonuniversal critical dynamics in Monte Carlo simulations. *Phys. Rev. Lett.*, 58:86–88, Jan 1987.

[26] J. Wakeley. *Coalescent Theory: An Introduction*. Roberts & Company Publishers, 1 edition, June 2008.

[27] B. Kirkpatrick, E. Halperin, and R. M. Karp. Haplotype inference in complex pedigrees. *Journal of Computational Biology*, 17(3):269–280, 2010.

[28] C. A. Albers, T. Heskes, and H. J. Kappen. Haplotype inference in general pedigrees using the cluster variation method. *Genetics*, 177(2):1101–1116, October 2007.

[29] The International HapMap Consortium. The international HapMap project. *Nature*, 426:789–796, 2003.

